# Multiple Incremental Decremental Learning of Support Vector Machines

**Masayuki Karasuyama and Ichiro Takeuchi**
Department of Engineering, Nagoya Institute of Technology
Gokiso-cho, Syouwa-ku, Nagoya, Aichi, 466-8555, JAPAN
krsym@ics.nitech.ac.jp, takeuchi.ichiro@nitech.ac.jp

## Abstract

We propose a multiple incremental decremental algorithm of Support Vector Machine (SVM). Conventional single incremental decremental SVM can update the trained model efficiently when single data point is added to or removed from the training set. When we add and/or remove multiple data points, this algorithm is time-consuming because we need to repeatedly apply it to each data point. The proposed algorithm is computationally more efficient when multiple data points are added and/or removed simultaneously. The single incremental decremental algorithm is built on an optimization technique called *parametric programming*. We extend the idea and introduce *multi-parametric programming* for developing the proposed algorithm. Experimental results on synthetic and real data sets indicate that the proposed algorithm can significantly reduce the computational cost of multiple incremental decremental operation. Our approach is especially useful for online SVM learning in which we need to remove old data points and add new data points in a short amount of time.

## 1 Introduction

Incremental decremental algorithm for online learning of Support Vector Machine (SVM) was previously proposed in [1], and the approach was adapted to other variants of kernel machines [2–4]. When a single data point is added and/or removed, these algorithms can efficiently update the trained model without re-training it from scratch. These algorithms are built on an optimization technique called *parametric programming* [5–7], in which one solves a series of optimization problems parametrized by a single parameter. In particular, one solves a solution path with respect to the coefficient parameter corresponding to the data point to be added or removed. When we add and/or remove multiple data points using these algorithms, one must repeat the updating operation for each single data point. It often requires too much computational cost to use it for real-time online learning. In what follows, we refer this conventional algorithm as *single incremental decremental algorithm* or *single update algorithm*.

In this paper, we develop a multiple incremental decremental algorithm of the SVM. The proposed algorithm can update the trained model more efficiently when multiple data points are added and/or removed simultaneously. We develop the algorithm by introducing *multi-parametric programming* [8] in the optimization literature. We consider a path-following problem in the multi-dimensional space spanned by the coefficient parameters corresponding to the set of data points to be added or removed. Later, we call our proposed algorithm as *multiple incremental decremental algorithm* or *multiple update algorithm*.

The main computational cost of parametric programming is in solving a linear system at each *breakpoint* (see Section 3 for detail). Thus, the total computational cost of parametric programming is roughly proportional to the number of breakpoints on the solution path. In the repeated use of

single update algorithm for each data point, one follows the coordinate-wise solution path in the multi-dimensional coefficient parameter space. On the other hand, in multiple update algorithm, we establish a direction in the multi-dimensional coefficient parameter space so that the total length of the path becomes much shorter than the coordinate-wise one. Because the number of breakpoints in the shorter path followed by our algorithm is less than that in the longer coordinate-wise path, we can gain relative computational efficiency. Figure 2 in Section 3.4 schematically illustrates our main idea.

This paper is organized as follows. Section 2 formulates the SVM and the KKT conditions. In Section 3, after briefly reviewing single update algorithm, we describe our multiple update algorithm. In section 4, we compare the computational cost of our multiple update algorithm with the single update algorithm and with the LIBSVM (the-state-of-the-art batch SVM solver based on SMO algorithm) in numerical experiments on synthetic and real data sets. We close in Section 5 with concluding remarks.

## 2  Support Vector Machine and KKT Conditions

Suppose we have a set of training data $\{(\boldsymbol{x}_i, y_i)\}_{i=1}^n$, where $\boldsymbol{x}_i \in \mathcal{X} \subseteq \mathbb{R}^d$ is the input and $y_i \in \{-1, +1\}$ is the output class label. Support Vector Machines (SVM) learn the following discriminant function:

$$f(\boldsymbol{x}) = \boldsymbol{w}^T \Phi(\boldsymbol{x}) + b,$$

where $\Phi(\boldsymbol{x})$ denotes a fixed feature-space transformation. The model parameter $\boldsymbol{w}$ and $b$ can be obtained by solving an optimization problem:

$$\min \quad \frac{1}{2}||\boldsymbol{w}||^2 + C\sum_{i=1}^n \xi_i$$

$$\text{s.t.} \quad y_i f(\boldsymbol{x}_i) \geq 1 - \xi_i, \ \xi_i \geq 0, \ i = 1, \cdots, n,$$

where $C \in \mathbb{R}^+$ is the regularization parameter. Introducing Lagrange multipliers $\alpha_i \geq 0$, the optimal discriminant function $f : \mathcal{X} \to \mathbb{R}$ can be formulated as $f(\boldsymbol{x}) = \sum_{i=1}^n \alpha_i y_i K(\boldsymbol{x}, \boldsymbol{x}_i) + b$, where $K(\boldsymbol{x}_i, \boldsymbol{x}_j) = \Phi(\boldsymbol{x}_i)^T \Phi(\boldsymbol{x}_j)$ is a kernel function. From the Karush-Kuhn-Tucker (KKT) optimality conditions, we obtain the following relationships:

$$y_i f(\boldsymbol{x}_i) > 1 \quad \Rightarrow \quad \alpha_i = 0, \tag{1a}$$

$$y_i f(\boldsymbol{x}_i) = 1 \quad \Rightarrow \quad \alpha_i \in [0, C], \tag{1b}$$

$$y_i f(\boldsymbol{x}_i) < 1 \quad \Rightarrow \quad \alpha_i = C, \tag{1c}$$

$$\sum_{i=1}^n y_i \alpha_i = 0. \tag{1d}$$

Using (1a)-(1c), let us define the following index sets:

$$\mathcal{O} \quad = \quad \{i \mid y_i f(\boldsymbol{x}_i) > 1, \alpha_i = 0\}, \tag{2a}$$

$$\mathcal{M} \quad = \quad \{i \mid y_i f(\boldsymbol{x}_i) = 1, 0 \leq \alpha_i \leq C\}, \tag{2b}$$

$$\mathcal{I} \quad = \quad \{i \mid y_i f(\boldsymbol{x}_i) < 1, \alpha_i = C\}. \tag{2c}$$

In what follows, the subscription by an index set, such as $\boldsymbol{v}_{\mathcal{I}}$ for a vector $\boldsymbol{v} \in \mathbb{R}^n$, indicates a subvector of $\boldsymbol{v}$ whose elements are indexed by $\mathcal{I}$. Similarly, the subscription by two index sets, such as $\boldsymbol{M}_{\mathcal{M}, \mathcal{O}}$ for a matrix $\boldsymbol{M} \in \mathbb{R}^{n \times n}$, denotes a submatrix whose rows are indexed by $\mathcal{M}$ and columns are indexed by $\mathcal{O}$. If the submatrix is the principal submatrix such as $\boldsymbol{Q}_{\mathcal{M}, \mathcal{M}}$, we abbreviate as $\boldsymbol{Q}_{\mathcal{M}}$.

## 3  Incremental Decremental Learning for SVM

### 3.1  Single Incremental Decremental SVM

In this section, we briefly review the conventional single incremental decremental SVM [1]. Using the SV sets (2b) and (2c), we can expand $y_i f(\boldsymbol{x}_i)$ as

$$y_i f(\boldsymbol{x}_i) = \sum_{j \in \mathcal{M}} Q_{ij} \alpha_j + \sum_{j \in \mathcal{I}} Q_{ij} \alpha_j + y_i b,$$

where $Q_{ij} = y_i y_j K(\boldsymbol{x}_i, \boldsymbol{x}_j)$. When a new data point $(\boldsymbol{x}_c, y_c)$ is added, we increase the corresponding new parameter $\alpha_c$ from 0 while keeping the optimal conditions of the other parameters satisfied. Let us denote the amount of the change of each variable with an operator $\Delta$. To satisfy the equality conditions (1b) and (1d), we need

$$Q_{ic}\Delta\alpha_c + \sum_{j \in \mathcal{M}} Q_{ij}\Delta\alpha_j + y_i\Delta b \;\; = \;\; 0, \; i \in \mathcal{M},$$

$$y_c\Delta\alpha_c + \sum_{j \in \mathcal{M}} y_j\Delta\alpha_j \;\; = \;\; 0.$$

Solving this linear system with respect to $\Delta\alpha_i, i \in \mathcal{M}$, and $b$, we obtain the update direction of the parameters. We maximize the $\Delta\alpha_c$ under the constraint that no element moves across $\mathcal{M}, \mathcal{I}$ and $\mathcal{O}$. After updating the index sets $\mathcal{M}, \mathcal{I}$ and $\mathcal{O}$, we repeat the process until the new data point satisfies the optimality condition. Decremental algorithm can be derived similarly, in which the target parameter moves toward 0.

## 3.2 Multiple Incremental Decremental SVM

Suppose we add $m$ new data points and remove $\ell$ data points simultaneously. Let us denote the index set of new adding data points and removing data points as

$$\mathcal{A} = \{n+1, n+2, \cdots, n+m\} \text{ and } \mathcal{R} \subset \{1, \cdots, n\},$$

respectively, where $|\mathcal{R}| = \ell$. We remove the elements of $\mathcal{R}$ from the sets $\mathcal{M}, \mathcal{I}$ and $\mathcal{O}$ (i.e. $\mathcal{M} \leftarrow \mathcal{M} \setminus \mathcal{R}, \mathcal{I} \leftarrow \mathcal{I} \setminus \mathcal{R}$ and $\mathcal{O} \leftarrow \mathcal{O} \setminus \mathcal{R}$). Let us define $\boldsymbol{y} = [y_1, \cdots, y_{n+m}]^\top, \boldsymbol{\alpha} = [\alpha_1, \cdots, \alpha_{n+m}]^\top$, and $\boldsymbol{Q} \in \mathbb{R}^{(n+m) \times (n+m)}$, where $(i, j)$-th entry of $\boldsymbol{Q}$ is $Q_{ij}$. When $m = 1, \ell = 0$ or $m = 0, \ell = 1$, our method corresponds to the conventional single incremental decremental algorithm. We initially set $\alpha_i = 0, \forall i \in \mathcal{A}$. If we have $y_i f(\boldsymbol{x}_i) > 1, i \in \mathcal{A}$, we can append these indices to $\mathcal{O}$ and remove them from $\mathcal{A}$ because these points already satisfy the optimality condition (1a). Similarly, we can append the indices $\{i \mid y_i f(\boldsymbol{x}_i) = 1, i \in \mathcal{A}\}$ to $\mathcal{M}$ and remove them from $\mathcal{A}$. In addition, we can remove the points $\{i \mid \alpha_i = 0, i \in \mathcal{R}\}$ because they already have no influence on the model. Unlike single incremental decremental algorithm, we need to determine the directions of $\boldsymbol{\Delta\alpha}_\mathcal{A}$ and $\boldsymbol{\Delta\alpha}_\mathcal{R}$. These directions have a critical influence on the computational cost. For $\boldsymbol{\Delta\alpha}_\mathcal{R}$, we simply trace the shortest path to $\boldsymbol{0}$, i.e.,

$$\boldsymbol{\Delta\alpha}_\mathcal{R} = -\eta\boldsymbol{\alpha}_\mathcal{R}, \tag{3}$$

where $\eta$ is a step length. For $\boldsymbol{\Delta\alpha}_\mathcal{A}$, we do not know the optimal value of $\boldsymbol{\alpha}_\mathcal{A}$ beforehand. To determine this direction, we may be able to use some optimization techniques (e.g. Newton method). However, such methods usually need additional computational burden. In this paper, we simply take

$$\boldsymbol{\Delta\alpha}_\mathcal{A} = \eta(C\boldsymbol{1} - \boldsymbol{\alpha}_\mathcal{A}). \tag{4}$$

This would become the shortest path if $\alpha_i = C, \forall i \in \mathcal{A}$, at optimality.

When we move parameters $\alpha_i, \forall i \in \mathcal{A} \cup \mathcal{R}$, the optimality conditions of the other parameters must be kept satisfied. From $y_i f(\boldsymbol{x}_i) = 1, i \in \mathcal{M}$, and the equality constraint (1d), we need

$$\sum_{j \in \mathcal{A}} Q_{ij}\Delta\alpha_j + \sum_{j \in \mathcal{R}} Q_{ij}\Delta\alpha_j + \sum_{j \in \mathcal{M}} Q_{ij}\Delta\alpha_j + y_i\Delta b \;\; = \;\; 0, \; i \in \mathcal{M}, \tag{5}$$

$$\sum_{j \in \mathcal{A}} y_j\Delta\alpha_j + \sum_{j \in \mathcal{R}} y_j\Delta\alpha_j + \sum_{j \in \mathcal{M}} y_j\Delta\alpha_j \;\; = \;\; 0. \tag{6}$$

Using matrix notation, (5) and (6) can be written as

$$\boldsymbol{M}\left[\begin{array}{c} \Delta b \\ \boldsymbol{\Delta\alpha}_\mathcal{M} \end{array}\right] + \left[\begin{array}{cc} \boldsymbol{y}_\mathcal{A}^\top & \boldsymbol{y}_\mathcal{R}^\top \\ \boldsymbol{Q}_{\mathcal{M},\mathcal{A}} & \boldsymbol{Q}_{\mathcal{M},\mathcal{R}} \end{array}\right]\left[\begin{array}{c} \boldsymbol{\Delta\alpha}_\mathcal{A} \\ \boldsymbol{\Delta\alpha}_\mathcal{R} \end{array}\right] = \boldsymbol{0}, \tag{7}$$

where

$$\boldsymbol{M} = \left[\begin{array}{cc} 0 & \boldsymbol{y}_\mathcal{M}^\top \\ \boldsymbol{y}_\mathcal{M} & \boldsymbol{Q}_\mathcal{M} \end{array}\right].$$

From the definitions of the index sets in (2a)-(2c), the following inequality constraints must also be satisfied:

$$0 \leq \alpha_i + \Delta\alpha_i \leq C, \qquad i \in \mathcal{M}, \tag{8a}$$

$$y_i\{f(\boldsymbol{x}_i) + \Delta f(\boldsymbol{x}_i)\} > 1, \qquad i \in \mathcal{O}, \tag{8b}$$

$$y_i\{f(\boldsymbol{x}_i) + \Delta f(\boldsymbol{x}_i)\} < 1, \qquad i \in \mathcal{I}. \tag{8c}$$

Since we removed the indices $\{i \mid f(\boldsymbol{x}_i) \geq 1\}$ from $\mathcal{A}$, we obtain

$$y_i\{f(\boldsymbol{x}_i) + \Delta f(\boldsymbol{x}_i)\} < 1, \qquad i \in \mathcal{A}. \tag{9}$$

During the process of moving $\alpha_i, i \in \mathcal{A}$, to $C$ from 0, if the inequality (9) becomes equality for any $i$, we can append the point to $\mathcal{M}$ and remove it from $\mathcal{A}$. On the other hand, if (9) holds until $\alpha_i$ becomes $C$, the point moves to $\mathcal{I}$. In the path following literature [8], the region that satisfies (8) and (9) is called *critical region (CR)*.

We decide update direction by the linear system (7) while monitoring inequalities (8) and (9). Substituting (3) and (4) to (7), we obtain the update direction

$$\begin{bmatrix} \Delta b \\ \boldsymbol{\Delta\alpha}_{\mathcal{M}} \end{bmatrix} = \eta\boldsymbol{\phi}, \text{ where } \boldsymbol{\phi} = -\boldsymbol{M}^{-1} \begin{bmatrix} \boldsymbol{y}_{\mathcal{A}}^{\top} & \boldsymbol{y}_{\mathcal{R}}^{\top} \\ \boldsymbol{Q}_{\mathcal{M},\mathcal{A}} & \boldsymbol{Q}_{\mathcal{M},\mathcal{R}} \end{bmatrix} \begin{bmatrix} C\boldsymbol{1} - \boldsymbol{\alpha}_{\mathcal{A}} \\ -\boldsymbol{\alpha}_{\mathcal{R}} \end{bmatrix}. \tag{10}$$

To determine step length $\eta$, we need to check inequalities (8) and (9). Using vector notation and the hadamard product $\odot$ (element-wise product [9]), we can write

$$\boldsymbol{y} \odot \boldsymbol{\Delta f} = \eta\,\boldsymbol{\psi}, \text{ where } \boldsymbol{\psi} = [\,\boldsymbol{y}\,\boldsymbol{Q}_{:,\mathcal{M}}\,]\,\boldsymbol{\phi} + \boldsymbol{Q}_{:,\mathcal{A}}(C\boldsymbol{1} - \boldsymbol{\alpha}_{\mathcal{A}}) - \boldsymbol{Q}_{:,\mathcal{R}}\boldsymbol{\alpha}_{\mathcal{R}}, \tag{11}$$

and the subscription ":" of $\boldsymbol{Q}$ denotes the index of all the elements $\{1, \cdots, n+m\}$. Since (10) and (11) are linear function of $\eta$, we can calculate the set of the largest step length $\eta$s for each $i$ at which the inequalities (8) and (9) becomes equality for $i$. The size of such $\eta$s is $|\mathcal{M}| \times 2 + |\mathcal{O}| + |\mathcal{I}| + |\mathcal{A}|$ and we define this set as $\mathcal{H}$. We determine the step length as follows:

$$\eta = \min(\{\tilde{\eta} \mid \tilde{\eta} \in \mathcal{H}, \ \tilde{\eta} \geq 0\} \cup \{1\}).$$

If $\eta$ becomes 1, we can terminate the algorithm because all the new data points in $\mathcal{A}$ and existing points in $\mathcal{M}, \mathcal{O}$ and $\mathcal{I}$ satisfy the optimality conditions and $\boldsymbol{\alpha}_{\mathcal{R}}$ is $\boldsymbol{0}$. Once we decide $\eta$, we can update $\boldsymbol{\alpha}_{\mathcal{M}}$ and $b$ using (10), and $\boldsymbol{\alpha}_{\mathcal{A}}$ and $\boldsymbol{\alpha}_{\mathcal{R}}$ using (3) and (4). In the path-following literature, the points at which the size of linear system (7) is changed are called *breakpoints*. If the $i$th data point reaches bound of any one of the constraints (8) and (9) we need to update $\mathcal{M}, \mathcal{O}$ and $\mathcal{I}$. After updating, we re-calculate $\boldsymbol{\phi}, \boldsymbol{\psi}$ to determine the next step length.

### 3.3 Empty Margin

We need to establish the way of dealing with the empty margin $\mathcal{M}$. In such case, we can not obtain the bias from $y_i f(\boldsymbol{x}_i) = 1, i \in \mathcal{M}$. Then we can only obtain the interval of the bias from

$$y_i f(\boldsymbol{x}_i) > 1, \qquad i \in \mathcal{O},$$
$$y_i f(\boldsymbol{x}_i) < 1, \qquad i \in \mathcal{I} \cup \mathcal{A}.$$

To keep these inequality constraints, the bias term must be in

$$\max_{i \in \mathcal{L}} y_i g_i \leq b \leq \min_{i \in \mathcal{U}} y_i g_i, \tag{12}$$

where

$$g_i = 1 - \sum_{i \in \mathcal{I}} \alpha_i Q_{ij} - \sum_{i \in \mathcal{A}} \alpha_i Q_{ij} - \sum_{i \in \mathcal{R}} \alpha_i Q_{ij},$$

and

$$\mathcal{L} = \{i \mid i \in \mathcal{O}, y_i = +1\} \cup \{i \mid i \in \mathcal{I} \cup \mathcal{A}, y_i = -1\},$$
$$\mathcal{U} = \{i \mid i \in \mathcal{O}, y_i = -1\} \cup \{i \mid i \in \mathcal{I} \cup \mathcal{A}, y_i = +1\}.$$

If this empty margin happens during the path-following, we look for the new data points which re-enter the margin. When the set $\mathcal{M}$ is empty, equality constraint (6) becomes

$$\sum_{i \in \mathcal{A}} y_i \Delta\alpha_i + \sum_{i \in \mathcal{R}} y_i \Delta\alpha_i = \eta\delta(\boldsymbol{\alpha}) = 0, \tag{13}$$

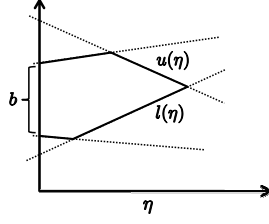

Figure 1: An illustration of the bias in empty margin case. Dotted lines represent $y_i(g_i + \Delta g_i(\eta))$, for each $i$. Solid lines are the upper bound and the lower bound of the bias. The bias term is uniquely determined when $u(\eta)$ and $l(\eta)$ intersect.

where

$$\delta(\boldsymbol{\alpha}) = \sum_{i \in \mathcal{A}} y_i(C - \alpha_i) - \sum_{i \in \mathcal{R}} y_i \alpha_i.$$

We take two different strategies depending on $\delta(\boldsymbol{\alpha})$.

First, if $\delta(\boldsymbol{\alpha}) \neq 0$, we can not simply increase $\eta$ from 0 while keeping (13) satisfied. Then we need new margin data point $m_1$ which enables equality constraint to be satisfied. The index $m_1$ is either

$$i_{low} = \operatorname*{argmax}_{i \in \mathcal{L}} y_i g_i \text{ or } i_{up} = \operatorname*{argmax}_{i \in \mathcal{U}} y_i g_i.$$

If $i_{low}, i_{up} \in \mathcal{O} \cup \mathcal{I}$, we can update $b$ and $\mathcal{M}$ as follows:

$$
\begin{aligned}
\delta(\boldsymbol{\alpha}) > 0 &\Rightarrow b = y_{i_{up}} g_{i_{up}}, \ \mathcal{M} = \{i_{up}\}, \\
\delta(\boldsymbol{\alpha}) < 0 &\Rightarrow b = y_{i_{low}} g_{i_{low}}, \ \mathcal{M} = \{i_{low}\}.
\end{aligned}
$$

By setting the bias terms as above, equality condition

$$\eta \delta(\boldsymbol{\alpha}) + y_{m_1} \Delta \alpha_{m_1} = 0$$

is satisfied. If $i_{low} \in \mathcal{A}$ or $i_{up} \in \mathcal{A}$, we can put either of these points to margin.

On the other hand, if $\delta(\boldsymbol{\alpha}) = 0$, we can increase $\eta$ while keeping (13) satisfied. Then, we consider increasing $\eta$ until the upper bound and the lower bound of the bias (12) take the same value (the bias term can be uniquely determined). If we increase $\eta$, $g_i$ changes linearly:

$$\Delta g_i(\eta) = -\sum_{j \in \mathcal{A}} \Delta \alpha_j Q_{ij} - \sum_{j \in \mathcal{R}} \Delta \alpha_j Q_{ij} = \eta \Big\{ -\sum_{j \in \mathcal{A}} (C - \alpha_j) Q_{ij} + \sum_{j \in \mathcal{R}} \alpha_j Q_{ij} \Big\}.$$

Since each $y_i(g_i + \Delta g_i(\eta))$ may intersect, we need to consider the following piece-wise linear boundaries:

$$
\begin{aligned}
u(\eta) &= \max_{i \in \mathcal{U}} y_i(g_i + \Delta g_i(\eta)), \\
l(\eta) &= \min_{j \in \mathcal{L}} y_j(g_j + \Delta g_j(\eta)).
\end{aligned}
$$

Figure 1 shows an illustration of these functions. We can trace the upper bound and the lower bound until two bounds become the same value.

### 3.4 The number of breakpoints

The main computational cost of incremental decremental algorithm is in solving the linear system (10) at each breakpoint (The cost is $O(|\mathcal{M}|^2)$ because we use Cholesky factor update except the first step). Thus, the number of breakpoints is an important factor of the computational cost. To simplify the discussion, let us introduce the following assumptions:

- The number of breakpoints is proportional to the total length of the path.
- The path obtained by our algorithm is the shortest one.

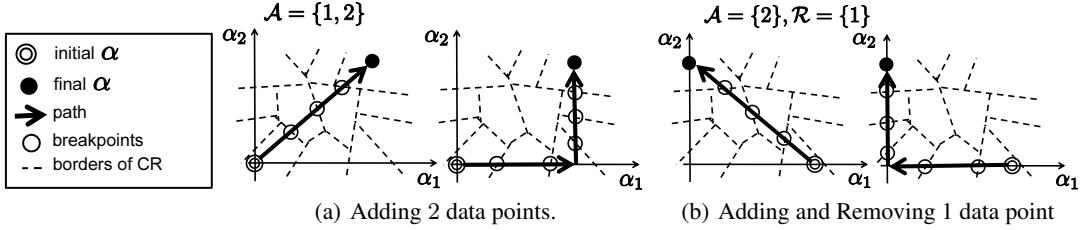

(a) Adding 2 data points.  (b) Adding and Removing 1 data point

Figure 2: The schematic illustration of the difference of path length and the number of breakpoints. Each polygonal region enclosed by dashed lines represents the region in which $\mathcal{M}, \mathcal{I}, \mathcal{O}$ and $\mathcal{A}$ are constant (CR: critical region). The intersection of the path and the borders are the breakpoints. The update of matrices and vectors at the breakpoints are the main computational cost of path-following. In the case of Figure 2(a), we add 2 data points. If optimal $\alpha_1 = \alpha_2 = C$, our proposed algorithm can trace shortest path to optimal point from the origin (left plot). On the other hand, single incremental algorithm moves one coordinate at a time (right plot). Figure 2(b) shows the case that we add and remove 1 data point, respectively. In this case, if $\alpha_2 = C$, our algorithm can trace shortest path to $\alpha_1 = 0, \alpha_2 = C$ (left plot), while single incremental algorithm again moves one coordinate at a time (right plot).

The first assumption means that the breakpoints are uniformly distributed on the path. The second assumption holds for the removing parameters $\boldsymbol{\alpha}_\mathcal{R}$ because we know that we should move $\boldsymbol{\alpha}_\mathcal{R}$ to $\mathbf{0}$. On the other hand, for some of $\boldsymbol{\alpha}_\mathcal{A}$, the second assumption does not necessarily hold because we do not know the optimal $\boldsymbol{\alpha}_\mathcal{A}$ beforehand. In particular, if the point $i \in \mathcal{A}$ which was located inside the margin before the update moved to $\mathcal{M}$ during the update (i.e. the equality (9) holds), the path with respect to this parameter is not really the shortest one.

To simplify the discussion further, let us consider only the case of $|\mathcal{A}| = m > 0$ and $|\mathcal{R}| = 0$ (the same discussion holds for other cases too). In this simplified scenario, the ratio of the number of breakpoints of multiple update algorithm to that of repeated use of single update algorithm is

$$\|\boldsymbol{\alpha}_\mathcal{A}\|_2 : \|\boldsymbol{\alpha}_\mathcal{A}\|_1,$$

where $\| \bullet \|_2$ is $\ell_2$ norm and $\| \bullet \|_1$ is $\ell_1$ norm. Figure 2 illustrates the concept in the case of $m = 2$. If we consider only the case of $\alpha_i = C, \forall i \in \mathcal{A}$, the ratio is simply $\sqrt{m} : m$.

## 4 Experiments

We compared the computational cost of the proposed multiple incremental decremental algorithm (MID-SVM) with (repeated use of) single incremental decremental algorithm [1] (SID-SVM) and with the LIBSVM [10], the-state-of-the-art batch SVM solver based on sequential minimal optimization algorithm (SMO).

In LIBSVM, we examined several tolerances for termination criterion: $\varepsilon = 10^{-3}, 10^{-6}, 10^{-9}$. When we use LIBSVM for online-learning, alpha seeding [11, 12] sometimes works well. The basic idea of alpha seeding is to use the parameters before the update as the initial parameter. In alpha seeding, we need to take care of the fact that the summation constraint $\boldsymbol{\alpha}^\top \boldsymbol{y} = 0$ may not be satisfied after removing $\alpha$s in $\mathcal{R}$. In that case, we simply re-distribute

$$\delta = \sum_{i \in \mathcal{R}} \alpha_i y_i$$

to the in-bound $\alpha_i, i \in \{i \mid 0 < \alpha_i < C\}$, uniformly. If $\delta$ cannot be distributed to in-bound $\alpha$s, it is also distributed to other $\alpha$s. If we still can not distribute $\delta$ by this way, we did not use alpha-seeding.

For kernel function, we used RBF kernel $K(\boldsymbol{x}_i, \boldsymbol{x}_j) = \exp(-\gamma \|\boldsymbol{x}_i - \boldsymbol{x}_j\|^2)$. In this paper, we assume that the kernel matrix $\boldsymbol{K}$ is positive definite. If the kernel matrix happens to be singular, which typically arise when there are two or more identical data points in $\mathcal{M}$, our algorithm may not work. As far as we know, this degeneracy problem is not fully solved in path-following literature. Many heuristics are proposed to circumvent the problem. In the experiments described below, we

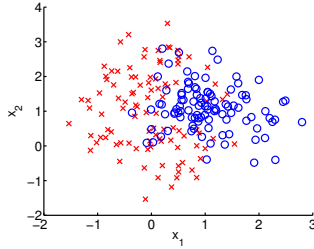

Figure 3: Artificial data set. For graphical simplicity, we plot only a part of data points. The cross points are generated from a mixture of two Gaussian while the circle points come from a single Gaussian. Two classes have equal prior probabilities.

use one of them: adding small positive constant to the diagonal elements of kernel matrix. We set this constant as $10^{-6}$. In the LIBSVM we can specify cache size of kernel matrix. We set this cache size enough large to store the entire matrix.

## 4.1 Artificial Data

First, we used simple artificial data set to see the computational cost for various number of adding and/or removing points. We generated data points $(\boldsymbol{x}, y) \in \mathbb{R}^2 \times \{+1, -1\}$ using normal distributions. Figure 3 shows the generated data points. The size of initial data points is $n = 500$. As discussed, adding or removing the data points with $\alpha_i = 0$ at optimal can be performed with almost no cost. Thus, to make clear comparison, we restrict the adding and/or removing points as those with $\alpha_i = C$ at optimal. Figure 4 shows the log plot of the CPU time. We examined several scenarios: (a) adding $m \in \{1, \cdots, 50\}$ data points, (b) removing $\ell \in \{1, \cdots, 50\}$ data points, (c) adding $m \in \{1, \cdots, 25\}$ data points and removing $\ell \in \{1, \cdots, 25\}$ data points simultaneously. The horizontal axis is the number of adding and/or removing data points. We see that MID-SVM is significantly faster than SID-SVM. When $m = 1$ or $\ell = 1$, SID-SVM and MID-SVM are identical. The relative difference of SID-SVM and MID-SVM grows as the $m$ and/or $\ell$ increase because MID-SVM can add or remove multiple data points simultaneously while SID-SVM merely iterates the algorithm $m + \ell$ times. In this experimental setting, the CPU time of SMO does not change largely because $m$ and $\ell$ are relatively smaller than $n$. Figure 5 shows the number of breakpoints of SID-SVM and MID-SVM along with the theoretical number of breakpoints of the MID-SVM in Section 3.4 (e.g., for scenario (a), the number of breakpoints of SID-SVM multiplied by $\sqrt{m}/m$). The results are very close to the theoretical one.

## 4.2 Application to Online Time Series Learning

We applied the proposed algorithm to a online time series learning problem, in which we update the model when some new observations arrive (adding the new ones and removing the obsolete ones). We used Fisher river data set in StatLib [13]. In this data set, the task is to predict whether the mean daily flow of the river increases or decreases using the previous 7 days temperature, precipitation and flow ($\boldsymbol{x}_i \in \mathbb{R}^{21}$). This data set contains the observations from Jan 1 1988 to Dec 31 1991. The size of the initial data points is $n = 1423$ and we set $m = \ell = 30$ (about a month). Each dimension of $\boldsymbol{x}$ is normalized to $[0, 1]$. We add new $m$ data points and remove the oldest $\ell$ data points. We investigate various settings of the regularization parameter $C \in \{10^{-1}, 10^0, \cdots, 10^5\}$ and kernel parameter $\gamma \in \{10^{-3}, 10^{-2}, 10^{-1}, 10^0\}$. Unlike previous experiments, we did not choose the adding or removing data points by its parameter. Figure 6 shows the elapsed CPU times and Figure 7 shows 10-fold cross-validation error of each setting. Each figure has 4 plots corresponding to different settings of kernel parameter $\gamma$. The horizontal axis denotes the regularization parameter $C$. Figure 6 shows that our algorithm is faster than the others, especially in large $C$. It is well known that the computational cost of SMO algorithm becomes large when $C$ gets large [14]. Cross-validation error in Figure 7 indicates that the relative computational cost of our proposed algorithm is especially low for the hyperparameters with good generalization performances in this application problem.

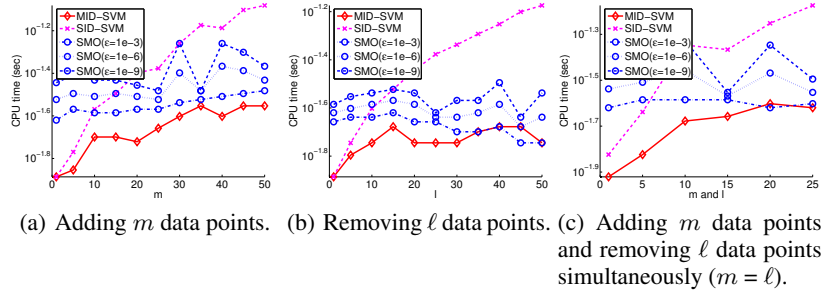

(a) Adding $m$ data points. (b) Removing $\ell$ data points. (c) Adding $m$ data points and removing $\ell$ data points simultaneously ($m = \ell$).

Figure 4: Log plot of the CPU time (artificial data set)

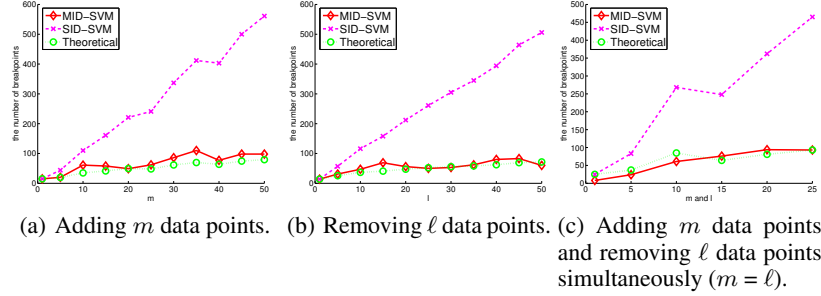

(a) Adding $m$ data points. (b) Removing $\ell$ data points. (c) Adding $m$ data points and removing $\ell$ data points simultaneously ($m = \ell$).

Figure 5: The number of breakpoints (artificial data set)

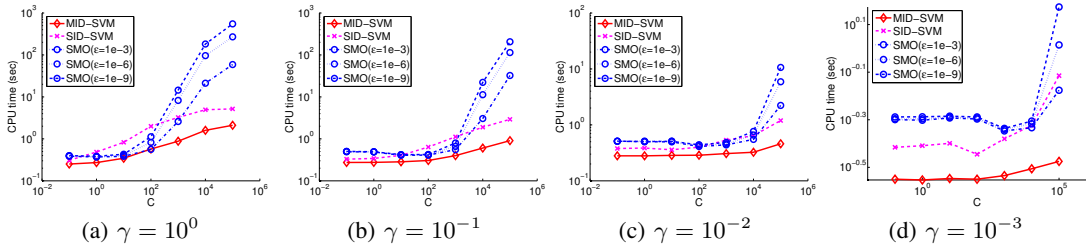

(a) $\gamma = 10^0$      (b) $\gamma = 10^{-1}$      (c) $\gamma = 10^{-2}$      (d) $\gamma = 10^{-3}$

Figure 6: Log plot of the CPU time (Fisher river data set)

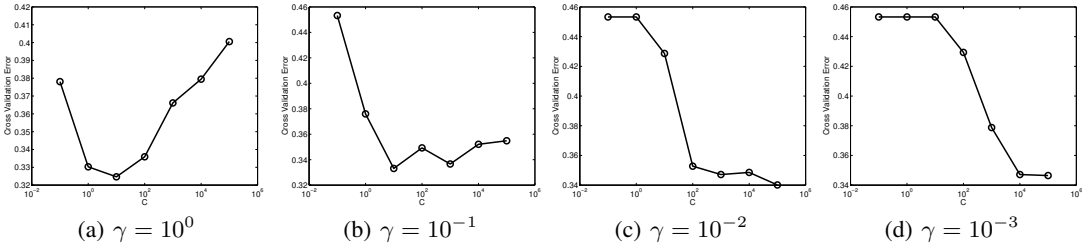

(a) $\gamma = 10^0$      (b) $\gamma = 10^{-1}$      (c) $\gamma = 10^{-2}$      (d) $\gamma = 10^{-3}$

Figure 7: Cross-validation error (Fisher river data set)

## 5 Conclusion

We proposed multiple incremental decremental algorithm of the SVM. Unlike single incremental decremental algorithm, our algorithm can efficiently work with simultaneous addition and/or removal of multiple data points. Our algorithm is built on multi-parametric programming in the optimization literature [8]. We previously proposed an approach to accelerate Support Vector Regression (SVR) cross-validation using similar technique [15]. These multi-parametric programming frameworks can be easily extended to other kernel machines.

# References

[1] G. Cauwenberghs and T. Poggio, "Incremental and decremental support vector machine learning," in *Advances in Neural Information Processing Systems* (T. K. Leen, T. G. Dietterich, and V. Tresp, eds.), vol. 13, (Cambridge, Massachussetts), pp. 409–415, The MIT Press, 2001.

[2] M. Martin, "On-line support vector machines for function approximation," tech. rep., Software Department, University Politecnica de Catalunya, 2002.

[3] J. Ma and J. Theiler, "Accurate online support vector regression," *Neural Computation*, vol. 15, no. 11, pp. 2683–2703, 2003.

[4] P. Laskov, C. Gehl, S. Kruger, and K.-R. Muller, "Incremental support vector learning: Analysis, implementation and applications," *Journal of Machine Learning Research*, vol. 7, pp. 1909–1936, 2006.

[5] T. Hastie, S. Rosset, R. Tibshirani, and J. Zhu, "The entire regularization path for the support vector machine," *Journal of Machine Learning Research*, vol. 5, pp. 1391–1415, 2004.

[6] L. Gunter and J. Zhu, "Efficient computation and model selection for the support vector regression," *Neural Computation*, vol. 19, no. 6, pp. 1633–1655, 2007.

[7] G. Wang, D.-Y. Yeung, and F. H. Lochovsky, "A new solution path algorithm in support vector regression," *IEEE Transactions on Neural Networks*, vol. 19, no. 10, pp. 1753–1767, 2008.

[8] E. N. Pistikopoulos, M. C. Georgiadis, and V. Dua, *Process Systems Engineering: Volume 1: Multi-Parametric Programming*. WILEY-VCH, 2007.

[9] J. R. Schott, *Matrix Analysis For Statistics*. Wiley-Interscience, 2005.

[10] C.-C. Chang and C.-J. Lin, "LIBSVM: a library for support vector machines," 2001. Software available at `http://www.csie.ntu.edu.tw/~cjlin/libsvm`.

[11] D. DeCoste and K. Wagstaff, "Alpha seeding for support vector machines," in *Proceedings of the International Conference on Knowledge Discovery and Data Mining*, pp. 345–359, 2000.

[12] M. M. Lee, S. S. Keerthi, C. J. Ong, and D. DeCoste, "An efficient method for computing leave-one-out error in support vector machines," *IEEE transaction on neural networks*, vol. 15, no. 3, pp. 750–757, 2004.

[13] M. Meyer, "Statlib." `http://lib.stat.cmu.edu/index.php`.

[14] L. Bottou and C.-J. Lin, "Support vector machine solvers," in *Large Scale Kernel Machines* (L. Bottou, O. Chapelle, D. DeCoste, and J. Weston, eds.), pp. 301–320, Cambridge, MA.: MIT Press, 2007.

[15] M. Karasuyama, I. Takeuchi, and R.Nakano, "Efficient leave-m-out cross-validation of support vector regression by generalizing decremental algorithm," *New Generation Computing*, vol. 27, no. 4, Special Issue on Data-Mining and Statistical Science, pp. 307–318, 2009.

